# Manifold Stochastic Dynamics
# for Bayesian Learning

**Mark Zlochin**
Department of Computer Science
Technion - Israel Institute of Technology
Technion City, Haifa 32000, Israel
zmark@cs.technion.ac.il

**Yoram Baram**
Department of Computer Science
Technion - Israel Institute of Technology
Technion City, Haifa 32000, Israel
baram@cs.technion.ac.il

## Abstract

We propose a new Markov Chain Monte Carlo algorithm which is a generalization of the stochastic dynamics method. The algorithm performs exploration of the state space using its intrinsic geometric structure, facilitating efficient sampling of complex distributions. Applied to Bayesian learning in neural networks, our algorithm was found to perform at least as well as the best state-of-the-art method while consuming considerably less time.

## 1  Introduction

In the Bayesian framework predictions are made by integrating the function of interest over the *posterior* parameter distribution, the latter being the normalized product of the *prior* distribution and the *likelihood*. Since in most problems the integrals are too complex to be calculated analytically, approximations are needed.

Early works in Bayesian learning for nonlinear models [Buntine and Weigend 1991, MacKay 1992] used Gaussian approximations to the posterior parameter distribution. However, the Gaussian approximation may be poor, especially for complex models, because of the multi-modal character of the posterior distribution.

Hybrid Monte Carlo (HMC) [Duane et al. 1987] introduced to the neural network community by [Neal 1996], deals more successfully with multi-modal distributions but is very time consuming. One of the main causes of HMC inefficiency is the anisotropic character of the posterior distribution - the density changes rapidly in some directions while remaining almost constant in others.

We present a novel algorithm which overcomes the above problem by using the intrinsic geometrical structure of the model space.

## 2  Hybrid Monte Carlo

Markov Chain Monte Carlo (MCMC) [Gilks et al. 1996] approximates the value

$$E[a] = \int a(\theta)Q(\theta)d\theta$$

by the mean

$$\bar{a} = \frac{1}{N} \sum_{t=1}^{N} a(\theta^{(t)})$$

where $\theta^{(1)}, \ldots, \theta^{(N)}$ are successive states of the ergodic Markov chain with invariant distribution $Q(\theta)$.

In addition to ergodicity and invariance of $Q(\theta)$ another quality we would like the Markov chain to have is rapid exploration of the state space. While the first two qualities are rather easily attained, achieving rapid exploration of the state space is often nontrivial.

A state-of-the-art MCMC method, capable of sampling from complex distributions, is Hybrid Monte Carlo [Duane et al. 1987].

The algorithm is expressed in terms of sampling from *canonical* distribution for the state, $q$, of a "physical" system, defined in terms of the energy function $E(q)$ [1]:

$$P(q) \propto \exp(-E(q)) \tag{1}$$

To allow the use of dynamical methods, a "momentum" variable, $p$, is introduced, with the same dimensionality as $q$. The canonical distribution over the "phase space" is defined to be:

$$P(q, p) \propto \exp(-H(q, p)) \tag{2}$$

where $H(q, p) = E(q) + K(p)$ is the "Hamiltonian", which represents the total energy. $K(p)$ is the "kinetic energy" due to momentum, defined as

$$K(p) = \sum_{i=1}^{n} \frac{p_i^2}{2m_i} \tag{3}$$

where $p_i, i = 1, \ldots, n$ are the momentum components and $m_i$ is the "mass" associated with $i$'th component, so that different components can be given different weight.

Sampling from the canonical distribution can be done using *stochastic dynamics* method [Andersen 1980], in which the task is split into two subtasks - sampling uniformly from values of $q$ and $p$ with a fixed total energy, $H(q, p)$, and sampling states with different values of $H$. The first task is done by simulating the *Hamiltonian dynamics* of the system:

$$\frac{dq_i}{d\tau} = +\frac{\partial H}{\partial p_i} = \frac{p_i}{m_i}$$

$$\frac{dp_i}{d\tau} = -\frac{\partial H}{\partial q_i} = -\frac{\partial E}{\partial q_i}$$

Different energy levels are obtained by occasional stochastic Gibbs sampling [Geman and Geman 1984] of the momentum. Since $q$ and $p$ are independent, $p$ may be updated without reference to $q$ by drawing a value with probability density proportional to $\exp(-K(p))$, which, in the case of (3), can be easily done, since the $p_i$'s have independent Gaussian distributions.

In practice, Hamiltonian dynamics cannot be simulated exactly, but can be approximated by some discretization using finite time steps. One common approximation is *leapfrog* discretization [Neal 1996].

In the hybrid Monte Carlo method stochastic dynamic transitions are used to generate candidate states for the Metropolis algorithm [Metropolis et al. 1953]. This eliminates certain

drawbacks of the stochastic dynamics such as systematic errors due to leapfrog discretization, since Metropolis algorithm ensures that every transition keeps canonical distribution invariant. However, the empirical comparison between the uncorrected stochastic dynamics and the HMC in application to Bayesian learning in neural networks [Neal 1996] showed that with appropriate discretization stepsize there is no notable difference between the two methods.

A modification proposed in [Horowitz 1991] instead of Gibbs sampling of momentum, is to replace $p$ each time by $p \cdot \cos(\theta) + \zeta \cdot \sin(\theta)$, where $\theta$ is a small angle and $\zeta$ is distributed according to $N(0, I)$. While keeping canonical distribution invariant, this scheme, called *momentum persistence*, improves the rate of exploration.

## 3  Riemannian geometry

A Riemannian manifold [Amari 1997] is a set $\Theta \subseteq R^n$ equipped with a *metric tensor $G$* which is a positive semidefinite matrix defining the inner product between infinitesimal increments as:

$$< d\theta_1, d\theta_2 >= d\theta_1^T \cdot G \cdot d\theta_2$$

Let us denote entries of $G$ by $G_{i,j}$ and entries of $G^{-1}$ by $G^{i,j}$. This inner product naturally gives us the norm

$$\| d\theta \|_G^2 =< d\theta, d\theta >= d\theta^T \cdot G \cdot d\theta.$$

The Jeffrey prior over $\Theta$ is defined by the density function:

$$\pi(\theta) \propto \sqrt{|G(\theta)|}$$

where $| \cdot |$ denotes determinant .

### 3.1  Hamiltonian dynamics over a manifold

For Riemannian manifold the dynamics take a more general form than the one described in section 2.

If the metric tensor is $G$ and all masses are set to one then the Hamiltonian is given by:

$$H(q, p) = E(q) + \frac{1}{2} p^T \cdot G^{-1} \cdot p \qquad (4)$$

The dynamics are governed by the following set of differential equations [Chavel 1993]:

$$\frac{d^2 q_i}{d\tau^2} = -\sum_j G^{i,j} \frac{\partial E}{\partial q_j} - \sum_{j,k} \Gamma^i_{j,k} \dot{q}_i \dot{q}_j$$

where $\Gamma^i_{j,k}$ are the *Christoffel symbols* given by:

$$\Gamma^i_{j,k} = \frac{1}{2} \sum_m G^{i,m} \left( \frac{\partial G_{m,k}}{\partial q_j} + \frac{\partial G_{m,j}}{\partial q_k} - \frac{\partial G_{j,k}}{\partial q_m} \right)$$

and $\dot{q} = \frac{dq}{d\tau}$ is related to $p$ by $\dot{q} = G^{-1} p$.

### 3.2 Riemannian geometry of functions

In regression the log-likelihood is proportional to the empirical error, which is simply the Euclidean distance between the target point, $t$, and candidate function evaluated over the sample. Therefore, the most natural distance measure between the models is the Euclidean seminorm :

$$d(\theta^1, \theta^2)^2 = \| f_{\theta^1} - f_{\theta^2} \|_l^2 = \sum_{i=1}^{l} (f(x_i, \theta^1) - f(x_i, \theta^2))^2 \tag{5}$$

The resulting metric tensor is:

$$G = \sum_{i=1}^{l} \{ \nabla_\theta f(x_i, \theta) \cdot \nabla_\theta f(x_i, \theta)^T \} = J^T \cdot J \tag{6}$$

where $\nabla_\theta$ denotes gradient and $J = [\dfrac{\partial f(x_i)}{\partial \theta_j}]$ is the Jacobian matrix.

### 3.3 Bayesian geometry

A Bayesian approach would suggest the inclusion of prior assumptions about the parameters in the manifold geometry.

If, for example, *a priori* $\theta \sim N(0, I/\alpha)$, then the log-posterior can be written as:

$$\log p(\theta|x) = \beta \sum_{i=1}^{l} (f(x_i, \theta_1) - t)^2 + \alpha \sum_{k=1}^{n} (\theta_k - 0)^2$$

where $\beta$ is inverse noise variance.

Therefore, the natural metric in the model space is

$$d(\theta^1, \theta^2)^2 = \beta \sum_{i=1}^{l} (f(x_i, \theta^1) - f(x_i, \theta^2))^2 + \alpha \sum_{k=1}^{n} (\theta_k^1 - \theta_k^2)^2$$

with the metric tensor:

$$G_B = \beta \cdot G + \alpha \cdot I = \hat{J}^T \cdot \hat{J} \tag{7}$$

where $\hat{J}$ is the "extended Jacobian":

$$\hat{J}_{i,j} = \begin{cases} \sqrt{\beta} \cdot \dfrac{\partial f(x_i)}{\partial \theta_j} & i \le l \\ \sqrt{\alpha} \cdot \delta_{i-l,j} & i > l \end{cases} \tag{8}$$

where $\delta_{i,j}$ is the Kroneker's delta.

Note, that as $\alpha \to 0$, $G_B \to \beta G$, hence as the prior becomes vaguer we approach a non-Bayesian paradigm. If, on the other hand, $\alpha \to \infty$ or $\beta \cdot G \to 0$, the Bayesian geometry approaches the Euclidean geometry of the parameter space. These are the qualities that we would like the Bayesian geometry to have - if the prior is "strong" in comparison to the likelihood, the exact form of $G$ should be of little importance.

The definitions above can be applied to any log-concave prior distribution with the inverse Hessian of the log-prior, $(\nabla\nabla \log p(\theta))^{-1}$, replacing $\alpha I$ in (7). The framework is not restricted to regression. For a general distribution class it is natural to use Fisher information matrix, $\mathcal{I}$, as a metric tensor [Amari 1997]. The Bayesian metric tensor then becomes:

$$G_B = \mathcal{I} + (\nabla\nabla \log p(\theta))^{-1} \tag{9}$$

## 4   Manifold Stochastic Dynamics

As mentioned before, the energy landscape in many regression problems is anisotropic. This degrades the performance of HMC in two aspects:

- The dynamics may not be optimal for efficient exploration of the posterior distribution as suggested by the studies of Gaussian diffusions [Hwang et al. 1993].
- The resulting differential equations are *stiff* [Gear 1971], leading to large discretization errors , which in turn necessitates small time steps, implying that the computational burden is high.

Both of these problems disappear if instead of the Euclidean Hamiltonian dynamics used in HMC we simulate dynamics over the manifold equipped with the metric tensor $G_B$ proposed in the previous section.

In the context of regression from the definition $G_B = \hat{J}^T \cdot \hat{J}$, we obtain an alternative equation for $\frac{d^2q}{d\tau^2}$ , in a matrix form:

$$\frac{d^2q}{d\tau^2} = -G_B^{-1}(\nabla E + \hat{J}^T \frac{\partial \hat{J}}{\partial \tau} \dot{q}) \tag{10}$$

In the canonical distribution $P(q,p) \propto \exp(-H(q,p))$ the conditional distribution of $p$ given $q$ is a zero-mean Gaussian with the covariance matrix $G_B(q)$ and the marginal distribution over $q$ is proportional to $\exp(-E(q))\pi(q)$. This is equivalent to multiplying the prior by the Jeffrey prior[2].

The sampling from the canonical distribution is two-fold:

- Simulate the Hamiltonian dynamics (3.1) for one time-step using leapfrog discretisation.
- Replace $p$ using momentum persistence. Unlike the HMC case, the momentum perturbation $\zeta$ is distributed according to $N(0, G_B)$.

The actual weights multiplying the matrices $I$ and $G$ in (7) may be chosen to be different from the specified $\alpha$ and $\beta$, so as to improve numerical stability.

## 5   Empirical comparison

### 5.1   Robot arm problem

We compared the performance of the Manifold Stochastic Dynamics (MSD) algorithm with the standard HMC. The comparison was carried using MacKay's robot arm problem which is a common benchmark for Bayesian methods in neural networks [MacKay 1992, Neal 1996].

The robot arm problem is concerned with the mapping :

$$y_1 = 2.0\cos x_1 + 1.3\cos(x_1 + x_2) + e_1, \quad y_2 = 2.0\sin x_1 + 1.3\sin(x_1 + x_2) + e_2$$

where $e_1, e_2$ are independent Gaussian noise variables of standard deviation 0.05. The dataset used by Neal and Mackay contained 200 examples in the training set and 400 in the test set.

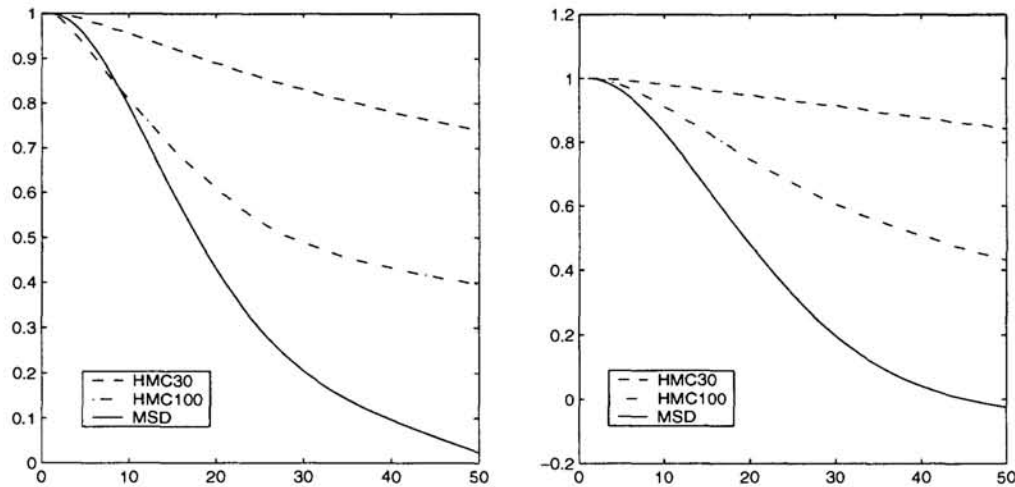

Figure 1: Average (over the 10 runs) autocorrelation of input-to-hidden (left) and hidden-to-output (right) weights for HMC with 100 and 30 leapfrog steps per iteration and MSD with single leapfrog step per iteration. The horizontal axis gives the lags, measured in number of iterations.

We used a neural network with two input units, one hidden layer containing 8 tanh units and two linear output units.

The hyperparameter $\beta$ was set to its correct value of 400 and $\alpha$ was chosen to be 1.

## 5.2 Algorithms

We compared MSD with two versions of HMC - with 30 and with 100 leapfrog steps per iteration, henceforth referred to as HMC30 and HMC100. MSD was run with a single leapfrog step per iteration. In all three algorithms momentum was resampled using persistence with $\cos(\theta) = 0.95$.

A single iteration of HMC100 required about $4.8 \cdot 10^6$ floating point operations (flops), HMC30 required $1.4 \cdot 10^6$ flops and MSD required $0.5 \cdot 10^6$ flops. Hence the computational load of MSD was about one third of that of HMC30 and 10 times lower than that of HMC100.

The discretization stepsize for HMC was chosen so as to keep the rejection rate below 5%. An equivalent criterion of average error in the Hamiltonian around 0.05 was used for the MSD .

All three sampling algorithms were run 10 times, each time for 3000 iteration with the first 1000 samples discarded in order to allow the algorithms to reach the regions of high probability.

## 5.3 Results

One appropriate measure for the rate of state space exploration is weights autocorrelation [Neal 1996]. As shown in Figure 1, the behavior of MSD was clearly superior to that of HMC.

Another value of interest is the total squared error over the test set. The predictions for the test set were made as follows. A subsample of 100 parameter vectors was generated by taking every twentieth sample vector starting from 1001 and on. The predicted value was

the average over the empirical function distribution of this subsample.

The total squared errors, normalized with respect to the variance on the test cases, have the following statistics (over the 10 runs):

|        | average | standard deviation |
|--------|---------|--------------------|
| HMC30  | 1.314   | 0.074              |
| HMC100 | 1.167   | 0.044              |
| MSD    | 1.161   | 0.023              |

The average error of HMC30 is high, indicating that the algorithm failed to reach the region of high probability. The errors of HMC100 and MSD are comparable but the standard deviation for MSD is twice as low as that for HMC100, meaning that the estimate obtained using MSD is more reliable.

## 6  Conclusion

We have described a new algorithm for efficient sampling from complex distributions such as those appearing in Bayesian learning with non-linear models. The empirical comparison shows that our algorithm achieves results superior to the best achieved by existing algorithms in considerably smaller computation time.

## Footnotes

[1]Note that any probability density that is nowhere zero can be put in this form, by simply defining $E(q) = -\log P(q) - \log Z$, for any convenient $Z$).

[2]In fact, since the actual prior over the weights is unknown, a truly Bayesian approach would be to use a non-informative prior such as $\pi(q)$. In this paper we kept the modified prior which is the product of $\pi(q)$ and a zero-mean Gaussian.

## References

[Amari 1997]      Amari S., "Natural Gradient Works Efficiently in Learning", *Neural Computation*, vol. 10, pp.251-276.

[Andersen 1980]   Andersen H.C., "Molecular dynamics simulations at constant pressure and/or temperature", *Journal of Chemical Physics*,vol. 3,pp. 589-603.

[Buntine and Weigend 1991] "Bayesian back-propagation", *Complex systems*, vol. 5, pp. 603-643.

[Chavel 1993]     Chavel I., *Riemannian Geometry: A Modern Introduction*, University Press, Cambridge.

[Duane et al. 1987] "Hybrid Monte Carlo", *Physics Letters B*,vol. 195,pp. 216-222.

[Gear 1971]       Gear C.W., *Numerical initial value problems in ordinary differential equations*, Prentice Hall.

[Geman and Geman 1984] Geman S.,Geman D., "Stochastic relaxation,Gibbs distributions and the Bayesian restoration of images", *IEEE Trans.,PAMI-6*,721-741.

[Gilks et al. 1996] Gilks W.R., Richardson S. and Spiegelhalter D.J., *Markov Chain Monte Carlo in Practice*, Chapman&Hall .

[Hwang et al. 1993] Hwang, C.,-R, Hwang-Ma S.,-Y. and Shen. S.,-J., "Accelerating Gaussian diffusions", *Ann. Appl. Prob.*, vol. 3, 897-913.

[Horowitz 1991]   Horowitz A.M., "A generalized guided Monte Carlo algorithm", *Physics Letters B,*, vol. 268, pp. 247-252.

[MacKay 1992]     MacKay D.J.C., *Bayesian Methods for Adaptive Models*, Ph.D. thesis, California Institute of Technology.

[Metropolis et al. 1953] Metropolis N., Rosenbluth A.W., Rosenbluth M.N., Teller A.H. and Teller E., "Equation of State Calculations by Fast Computing Machines", *Journal of Chemical Physics*,vol.21,pp. 1087-1092.

[Neal 1996]       Neal, R.M., *Bayesian Learning for Neural Networks*, Springer 1996.
